# Structural equations and divisive normalization for energy-dependent component analysis

**Jun-ichiro Hirayama**
Dept. of Systems Science
Graduate School of of Informatics
Kyoto University
611-0011 Uji, Kyoto, Japan

**Aapo Hyvärinen**
Dept. of Mathematics and Statistics
Dept. of Computer Science and HIIT
University of Helsinki
00560 Helsinki, Finland

## Abstract

Components estimated by independent component analysis and related methods are typically not independent in real data. A very common form of nonlinear dependency between the components is correlations in their variances or energies. Here, we propose a principled probabilistic model to model the energy-correlations between the latent variables. Our two-stage model includes a linear mixing of latent signals into the observed ones like in ICA. The main new feature is a model of the energy-correlations based on the structural equation model (SEM), in particular, a Linear Non-Gaussian SEM. The SEM is closely related to divisive normalization which effectively reduces energy correlation. Our new two-stage model enables estimation of both the linear mixing and the interactions related to energy-correlations, without resorting to approximations of the likelihood function or other non-principled approaches. We demonstrate the applicability of our method with synthetic dataset, natural images and brain signals.

## 1 Introduction

Statistical models of natural signals have provided a rich framework to describe how sensory neurons process and adapt to ecologically-valid stimuli [28, 12]. In early studies, independent component analysis (ICA) [2, 31, 13] and sparse coding [22] have successfully shown that V1 simple cell-like edge filters, or receptive fields, emerge as optimal *inference* on latent quantities under linear generative models trained on natural image patches. In the subsequent developments over the last decade, many studies (e.g. [10, 32, 11, 14, 23, 17]) have focused explicitly or implicitly on modeling a particular type of nonlinear dependency between the responses of the linear filters, namely correlations in their variances or *energies*. Some of them showed that models on energy-correlation could account for, e.g., response properties of V1 complex cells [10, 15], cortical topography [11, 23], and contrast gain control [26].

Interestingly, such energy correlations are also prominent in other kinds of data, including brain signals [33] and presumably even financial time series which have strong heteroscedasticity. Thus, developing a general model for energy-correlations of linear latent variables is an important problem in the theory of machine learning, and such models are likely to have a wide domain of applicability.

Here, we propose a new statistical model incorporating energy-correlations within the latent variables. Our two-stage model includes a linear mixing of latent signals into the observed ones like in ICA, and a model of the energy-correlations based on the structural equation model (SEM) [3], in particular the Linear Non-Gaussian (LiNG) SEM [27, 18] developed recently. As a model of natural signals, an important feature of our model is its connection to "divisive normalization" (DN) [7, 4, 26], which effectively reduces energy-correlations of linearly-transformed natural signals [32, 26, 29, 19, 21] and is now part of a well-accepted model of V1 single cell responses [12].

We provide a new *generative* interpretation of DN based on the SEM, which is an important contribution of this work. Also, from machine learning perspective, causal analysis by using SEM has recently become very popular; our model could extend the applicability of LiNG-SEM for blindly mixed signals.

As a two-stage extension of ICA, our model is also closely related to both the scale-mixture-based models, e.g. [11, 30, 14] (see also [32]) and the energy-based models, e.g. [23, 17]. An advantage of our new model is its tractability: our model requires neither an approximation of likelihood function nor non-canonical principles for modeling and estimation as previous models.

## 2    Structural equation model and divisive normalization

A structural equation model (SEM) [3] of a random vector $\boldsymbol{y} = (y_1, y_2, \ldots, y_d)^\top$ is formulated as simultaneous equations of random variables, such that

$$y_i = \kappa_i(y_i, \boldsymbol{y}_{-i}, r_i), \quad i = 1, 2, \ldots, d, \tag{1}$$

or $\boldsymbol{y} = \kappa(\boldsymbol{y}, \boldsymbol{r})$, where the function $\kappa_i$ describes how each single variable $y_i$ is related to other variables $\boldsymbol{y}_{-i}$, possibly including itself, and a corresponding stochastic *disturbance* or *external input* $r_i$ which is independent of $\boldsymbol{y}$. These equations, called structural equations, specify the distribution of $\boldsymbol{y}$, as $\boldsymbol{y}$ is an implicit function (assuming the system is invertible) of the random vector $\boldsymbol{r} = (r_1, r_2, \ldots, r_d)^\top$.

If there exists a permutation $\Pi : \boldsymbol{y} \mapsto \boldsymbol{y}'$ such that each $y_i'$ only depends on the preceding ones $\{y_j' | j < i\}$, an SEM is called recursive or acyclic, associated with a directed acyclic graph (DAG); the model is then a cascade of (possibly) nonlinear regressions of $y_i$'s on the preceding variables on the graph, and is also seen as a Bayesian network. Otherwise, the SEM is called non-recursive or cyclic, where the structural equations cannot be simply decomposed into regressive models. In a standard interpretation, a cyclic SEM rather describes the distribution of equilibrium points of a dynamical system, $\boldsymbol{y}(t) = \kappa(\boldsymbol{y}(t-1), \boldsymbol{r})$ $(t = 0, 1, \ldots)$, where every realized input $\boldsymbol{r}$ is fixed until $\boldsymbol{y}(t)$ converges to $\boldsymbol{y}$ [24, 18]; some conditions are usually needed to make the interpretation valid.

### 2.1    Divisive normalization as non-linear SEM

Now, we briefly point out the connection of SEM to DN, which strongly motivated us to explore the application of SEM to natural signal statistics.

Let $s_1, s_2, \ldots, s_d$ be scalar-valued outputs of $d$ linear filters applied to a multivariate input, collectively written as $\boldsymbol{s} = (s_1, s_2, \ldots, s_d)^\top$. The linear filters may either be derived/designed with some mathematical principles (e.g. Wavelets) or be learned from data (e.g. ICA). The outputs of linear filters often have the property that their *energies* $\phi(|s_i|)$ $(i = 1, 2, \ldots, d)$ have non-negligible dependencies or correlations to each other, even when the outputs themselves are linearly uncorrelated. The nonlinear function $\phi$ is any appropriate measure of energy, typically given by the squaring function, i.e. $\phi(|s|) = s^2$ [26, 12], while other choices will not be excluded; we assume $\phi$ is continuously differentiable and strictly increasing over $[0, \infty)$, and $\phi(0) = 0$.

Divisive Normalization (DN) [26] is an effective nonlinear transformation for eliminating the energy-dependencies remained in the filtered outputs. Although several variants have been proposed, a basic form can be formulated as follows: Given the $d$ outputs, their energies are normalized (divided) by a linear combination of the energies of other signals, such that

$$z_i = \frac{\phi(|s_i|)}{\sum_j h_{ij} \phi(|s_j|) + h_{i0}}, \quad i = 1, 2, \ldots, d, \tag{2}$$

where $h_{ij}$ and $h_{i0}$ are real-valued parameters of this transform. Now, it is straightforward to see that the following structural equations in the *log-energy* domain,

$$y_i := \ln \phi(|s_i|) = \ln(\sum_j h_{ij} \exp(y_j) + h_{i0}) + r_i, \quad i = 1, 2, \ldots, d, \tag{3}$$

correspond to Eq. (2) where $z_i = \exp(r_i)$ is another representation of the disturbance. The SEM will typically be cyclic, since the coefficients $h_{ij}$ in Eq. (2) are seldom constrained to satisfy acyclicity;

Eq. (3) thus implies a nonlinear dynamical system, and this can be interpreted as the data-generating processes underlying DN. Interestingly, Eq. (3) also implies a linear system with multiplicative input, $\widetilde{y}_i = (\sum_j h_{ij}\widetilde{y}_j + h_{i0})z_i$, in the energy domain, i.e. $\widetilde{y}_i := \phi(|s_i|)$. The DN transform of Eq. (2) gives the optimal mapping under the SEM to infer the disturbance from given $s_i$'s; if the true disturbances are independent, it optimally reduces the energy-dependencies. This is consistent with the redundancy reduction view of DN [29, 19].

Note also that the SEM above implies $\widetilde{y} = (\mathbf{I} - \mathrm{diag}(z)\mathbf{H})^{-1}\mathrm{diag}(h_0)z$ with $\mathbf{H} = (h_{ij})$ and $h_0 = (h_{i0})$, as shown in [20] in the context of DN [1]. Although mathematically equivalent, such a complicated dependence [20] on the disturbance $z$ does not provide an elegant model of the underlying data-generating process, compared to relatively the simple form of Eq. (3).

# 3 Energy-dependent ICA using structural equation model

Now, we define a new generative model which models energy-dependencies of linear latent components using an SEM.

## 3.1 Scale-mixture model

Let $s$ now be a random vector of $d$ source signals underlying an observation $x = (x_1, x_2, \ldots, x_d)^\top$ which has the same dimensionality for simplicity. They follow a standard linear generative model:

$$x = \mathbf{A}s, \tag{4}$$

where $\mathbf{A}$ is a square mixing matrix. We assume here $\mathrm{E}[x] = \mathrm{E}[s] = \mathbf{0}$ without loss of generality, by always subtracting the sample mean from every observation. Then, assuming $\mathbf{A}$ is invertible, each transposed row $w_i$ of the demixing (filtering) matrix $\mathbf{W} = \mathbf{A}^{-1}$ gives the optimal filter to recover $s_i$ from $x$, which is constrained to have unit norm, $\|w_i\|_2^2 = 1$ to fix the scaling ambiguity.

To introduce energy-correlations into the sources, a classic approach is to use a scale-mixture representation of sources, such that $s_i = u_i\sigma_i$, where $u_i$ represents a normalized signal having zero mean and constant variance, and $\sigma_i$ is a positive factor that is independent of $u_i$ and modulates the variance (energy) of $s_i$ [32, 11, 30, 14, 16]. Also, in vector notation, we write

$$s = u \odot \sigma, \tag{5}$$

where $\odot$ denotes component-wise multiplication. Here, $u$ and $\sigma$ are mutually independent, and $u_i$'s are also independent of each other. Then $\mathrm{E}[s|\sigma] = \mathbf{0}$ and $\mathrm{E}[ss^\top|\sigma] = \mathrm{diag}(\sigma_1^2, \sigma_2^2, \ldots, \sigma_d^2)$ for any given $\sigma$, where $\sigma_i$'s may be dependent of each other and introduce energy-correlations. A drawback of this approach is that to learn effectively the model based on the likelihood, we usually need some approximation to deal with the marginalization over $u$.

## 3.2 Linear Non-Gaussian SEM

Here, we simplify the above scale-mixture model by restricting $u_i$ to be binary, i.e. $u_i \in \{-1, 1\}$, and uniformly distributed. Although the simplification reduces the flexibility of source distribution, the resultant model is tractable, i.e. no approximation is needed for likelihood computation, as will be shown below. Also, this implies that $u_i = \mathrm{sign}(s_i)$ and $\sigma_i = |s_i|$, and hence the log-energy above now has a simple deterministic relation to $\sigma_i$, i.e. $y_i = \ln\phi(\sigma_i)$, which can be inverted to $\sigma_i = \phi^{-1}(\exp(y_i))$.

We particularly assume the log-energies $y_i$ follow the Linear Non-Gaussian (LiNG) [27, 18] SEM:

$$y_i = \sum_j h_{ij}y_j + h_{i0} + r_i, \quad i = 1, 2, \ldots, d, \tag{6}$$

where the disturbances are zero-mean and in particular assumed to be non-Gaussian and independent of each other, which has been shown to greatly improve the identifiability of linear SEMs [27]; the interaction structure in Eq. (6) can be represented by a directed graph for which the matrix

$\mathbf{H} = (h_{ij})$ serves as the weighted adjacency matrix. In the energy domain, Eq. (6) is equivalent to $\widetilde{y}_i = \left( \prod_j \widetilde{y}_j^{h_{ij}} \right) e^{h_{i0}} z_i$ $(i = 1, 2, \ldots, d)$, and interestingly, these SEMs further imply a novel form of DN transform, given by

$$z_i = \frac{\phi(|s_i|)}{e^{h_{i0}} \prod_j \phi(|s_j|)^{h_{ij}}}, \quad i = 1, 2, \ldots, d, \tag{7}$$

where the denominator is now not additive but multiplicative. It provides an interesting alternative to the original DN.

To recapitulate the new generative model proposed here: 1) The log-energies $\boldsymbol{y}$ are generated according to the SEM in Eq. (6); 2) the sources are generated according to Eq. (5) with $\sigma_i = \phi^{-1}(\exp(y_i))$ and random signs, $u_i$; and 3) the observation $\boldsymbol{x}$ is obtained by linearly mixing the sources as in Eq. (4). In our model, the optimal mapping to infer $z_i = \exp(r_i)$ from $\boldsymbol{x}$ under this model is the linear filtering $\mathbf{W}$ followed by the new DN transform, Eq. (7). On the other hand, it would also be possible to define the energy-dependent ICA by using the nonlinear SEM in Eq. (3) instead. Then, the optimal inference would be given by the divisive normalization in Eq. (2). However, estimation and other theoretical issues (e.g. identifiability) related to nonlinear SEMs, particularly in the case of non-Gaussianity of the disturbances, are quite involved, and are still under development, e.g. [8].

### 3.3 Identifiability issues

Both the theory and algorithms related to LiNG coincide largely with those of ICA, since Eq. (6) with non-Gaussian $\boldsymbol{r}$ implies the generative model of ICA, $\boldsymbol{y} = \mathbf{B}\boldsymbol{r} + \boldsymbol{b}_0$, where $\mathbf{B} = (\mathbf{I} - \mathbf{H})^{-1}$ and $\boldsymbol{b}_0 = \mathbf{B}\boldsymbol{h}_0$ with $\boldsymbol{h}_0 = (h_{i0})$. Like ICA [13], Eq. (6) is not completely identifiable due to the ambiguities related to scaling (with signs) and permutation [27, 18]. To fix the scaling, we set $\mathrm{E}[\boldsymbol{r}\boldsymbol{r}^\top] = \mathbf{I}$ here. The permutation ambiguity is more serious than in the case of ICA, because the row-permutation of $\mathbf{H}$ completely changes the structure of corresponding directed graph, and is typically addressed by constraining the graph structure, as will be discussed next.

Two classes of LiNG-SEM have been proposed, corresponding to different constraints on the graph structure. One is LiNGAM [27], which ensures the full identifiability by the DAG constraint. The other is generally referred to as LiNG [18] which allows general cyclic graphs; the "LiNG discovery" algorithm in [18] dealt with the non-identifiability of cyclic SEMs by finding out multiple solutions that give the same distribution.

Here we define two variants of our model: One is the *acyclic model*, using LiNGAM. In contrast to original LiNGAM, our target is (linear) latent variables, but not observed ones. The ordering of latent variables is not meaningful, because the rows of filter matrix $\mathbf{W}$ can be arbitrarily permuted. The acyclic constraint thus can be simplified into a lower-triangular constraint on $\mathbf{H}$. Another one is the *symmetric model*, which uses a special case of cyclic SEM, i.e. those with a symmetric constraint on $\mathbf{H}$. Such constraint would be relatively new to the context of SEM, although it is a well-known setting in the ICA literature (e.g. [5]). The SEM is then identifiable using only the first- and second-order statistics, based on the relations $\boldsymbol{h}_0 = \mathbf{V}\mathrm{E}[\boldsymbol{y}]$ and $\mathbf{V} := \mathbf{I} - \mathbf{H} = \mathrm{Cov}[\boldsymbol{y}]^{-\frac{1}{2}}$ [5], provided that $\mathbf{V}$ is positive definite [2]. This implies the non-Gaussianity is not essential for identifiability, in contrast that the acyclic model is not identifiable without non-Gaussianity [27]. The above relations also suggest moment-based estimators of $\boldsymbol{h}_0$ and $\mathbf{V}$, which can be used either as the final estimates or as the initial conditions in the maximum likelihood algorithm below.

### 3.4 Maximum likelihood

Let $\psi(s) := \ln \phi(|s|)$ for notational simplicity, and denote $\psi'(s) := \mathrm{sign}(s)(\ln \phi)'(|s|)$ as a convention, e.g. $(\ln |s|)' := 1/s$. Also, following the basic theory of ICA, we assume the disturbances have a joint probability density function (pdf) $p_{\boldsymbol{r}}(\boldsymbol{r}) = \prod_i \rho(r_i)$ with a common fixed marginal pdf $\rho$. Then, we have the following pdf of $\boldsymbol{s}$ without any approximation (see Appendix for derivation):

$$p_{\boldsymbol{s}}(\boldsymbol{s}) = \frac{1}{2^d} |\det \mathbf{V}| \prod_{i=1}^d \rho(\boldsymbol{v}_i^\top \psi(\boldsymbol{s}) - h_{i0}) |\psi'(s_i)|. \tag{8}$$

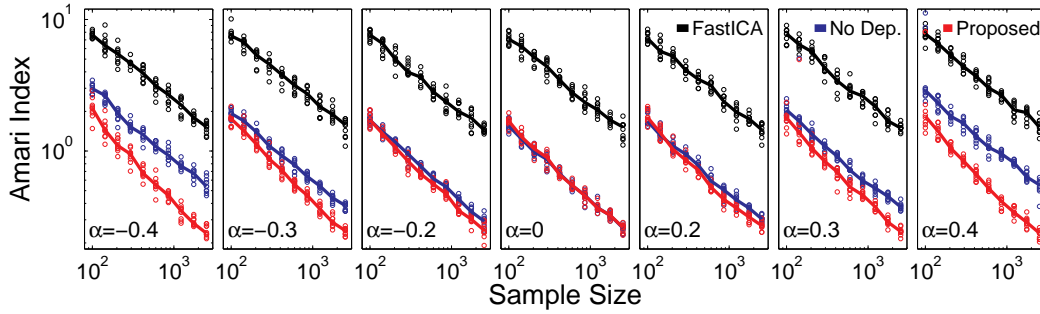

Figure 1: Estimation performance of mixing matrix measured by the "Amari Index" [1] (non-negative, and zero denotes perfect estimation up to unavoidable indeterminacies) versus sample size, shown in log-log scales. Each panel corresponds to a particular value of $\alpha$, which determined the relative connection strength between sources. The solid lines denotes the median of ten runs.

where $\boldsymbol{v}_i$ is $i$-th transposed row vector of $\mathbf{V}$ $(= \mathbf{I} - \mathbf{H})$. The pdf of $\boldsymbol{x}$ is given by $p_{\boldsymbol{x}}(\boldsymbol{x}) = |\det \mathbf{W}| p_{\boldsymbol{s}}(\mathbf{W}\boldsymbol{x})$, and the corresponding loss function, $l = -\ln p_{\boldsymbol{x}}(\boldsymbol{x}) + \text{const.}$, is given by

$$l(\boldsymbol{x}, \mathbf{W}, \mathbf{V}, \boldsymbol{h}_0) = \bar{f}(\mathbf{V}\psi(\mathbf{W}\boldsymbol{x}) - \boldsymbol{h}_0) + \bar{g}(\mathbf{W}\boldsymbol{x}) - \ln|\det \mathbf{W}| - \ln|\det \mathbf{V}|, \qquad (9)$$

where $\bar{f}(\boldsymbol{r}) = \sum_i f(r_i)$, $f(r_i) = -\ln \rho(r_i)$, $\bar{g}(\boldsymbol{s}) = \sum_i g(s_i)$, and $g(s_i) = -\ln|\psi'(s_i)|$.

Note that the loss function above is closely related to the ones in previous studies, such as of energy-based models [23, 17]. Our model is less flexible to these models, since it is limited to the case that $\mathbf{A}$ is square, but the exact likelihood is available. It is also interesting to see that the loss function above includes an additional second term that has not appeared in previous models, due to the formal derivation of pdf by the argument of transformation of random variables.

To obtain the maximum likelihood estimates of $\mathbf{W}$, $\mathbf{V}$, and $\boldsymbol{h}_0$, we minimize the negative log-likelihood (i.e. empirical average of the losses) by the projected gradient method (for the unit-norm constraints, $\|\boldsymbol{w}_i\|_2^2 = 1$). The required first derivatives are given by

$$\frac{\partial l}{\partial \boldsymbol{h}_0} = -f'(\boldsymbol{r}), \quad \frac{\partial l}{\partial \mathbf{V}} = f'(\mathbf{V}\boldsymbol{y} - \boldsymbol{h}_0)\boldsymbol{y}^\top - \mathbf{V}^{-\top}, \qquad (10a)$$

$$\frac{\partial l}{\partial \mathbf{W}} = \left\{ \text{diag}(\psi'(\mathbf{W}\boldsymbol{x}))\mathbf{V}^\top f'(\mathbf{V}\boldsymbol{y} - \boldsymbol{h}_0) + g'(\mathbf{W}\boldsymbol{x}) \right\} \boldsymbol{x}^\top - \mathbf{W}^{-\top}. \qquad (10b)$$

In both acyclic and symmetric cases, only the lower-triangular elements in $\mathbf{V}$ are free parameters. If acyclic, the upper-triangular elements are fixed at zero; if symmetric, they are dependent of the lower-triangular elements, and thus $\partial l/\partial v_{ij}$ $(i > j)$ should be replaced with $\partial l/\partial v_{ij} + \partial l/\partial v_{ji}$.

## 4    Simulations

To demonstrate the applicability of our method, we conducted the following simulation experiments. In all experiments below, we set $\phi(|s|) = |s|$, and $\rho(r) = (1/2)\text{sech}(\pi r/2)$ corresponding to the standard $\tanh$ nonlinearity in ICA: $f'(r) = (\pi/2)\tanh((\pi/2)r)$. In our projected gradient algorithm, the matrix $\mathbf{W}$ was first initialized by FastICA [9]; the SEM parameters, $\mathbf{H}$ and $\boldsymbol{h}_0$, were initialized by the moment-based estimator described above (symmetric model) or by the LiNGAM algorithm [27] (acyclic model). The algorithm was terminated when the decrease of objective value was smaller than $10^{-6}$; the learning rate was adjusted in each step by simply multiplying it by the factor $0.9$ until the new point did not increase the objective value.

### 4.1    Synthetic dataset

First, we examined how the energy-dependence learned in the SEM affects the estimation of linear filters. We artificially sampled the dataset with $d = 10$ from our generative model by setting the matrix $\mathbf{V}$ to be tridiagonal, where all the main and the first diagonals were set at $10$ and $10\alpha$, respectively. Figure 1 shows the "Amari Index" [1] of estimated $\mathbf{W}$ by three methods, at several

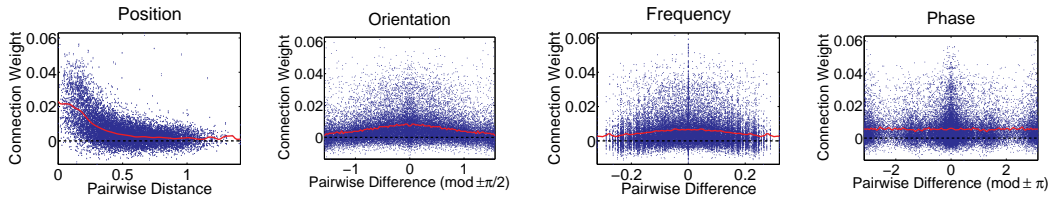

Figure 2: Connection weights versus pairwise differences of four properties of linear basis functions, estimated by fitting 2D Gabor functions. The curves were fit by local Gaussian smoothing.

factors $\alpha$ and sample sizes, with ten runs for every condition. In each run, the true mixing matrix was given by inverting $\mathbf{W}$ randomly generated from standard Gaussian and then row-normalized to have unit norms. The three methods were: 1) FastICA [3] with the $\mathrm{tanh}$ nonlinearity, 2) Our method (symmetric model) without energy-dependence (NoDep) initialized by FastICA, and 3) Our full method (symmetric model) initialized by NoDep. NoDep was the same as the full method except that the off-diagonal elements of $\mathbf{H}$ was kept zero. Note that our two algorithms used exactly the same criterion for termination of algorithm, while FastICA used a different one. This could cause the relatively poor performance of FastICA in this figure. The comparison between the full method and NoDep showed that energy-dependence learned in the SEM could improve the estimation of filter matrix, especially when the dependence was relatively strong.

### 4.2 Natural images

The dataset consisted of $50,000$ image patches of $16 \times 16$ pixels randomly taken from the original gray-scale pictures of natural scenes [4]. As a preprocessing, the sample mean was subtracted and the dimensionality was reduced to $160$ by the principal component analysis (PCA) where $99\%$ of the variance was retained. We constrained the SEM to be symmetric. Both of the obtained basis functions and filters were qualitatively very similar to those reported in many previous studies, and given in the Supplementary Material.

Figure 2 shows the values of connection weights $h_{ij}$ (after a row-wise re-scaling of $\mathbf{V}$ to set any $h_{ii} = 1 - v_{ii}$ to be zero, as a standard convention in SEM [18]) for every $d(d-1)$ pairs, compared with the pairwise difference of four properties of learned features (i.e. basis functions), estimated by fitting 2D Gabor functions: spatial positions, frequencies, orientations and phases. As is clearly seen, the connection weights tended to be large if the features were similar to each other, except for their phases; the phases were not strongly correlated with the weights as suggested by the fitted curve, while they exhibited a weak tendency to be the same or the opposite (shifted $\pm\pi$) to each other. We can also see a weak tendency for the negative weights to have large magnitudes when the pairs have near-orthogonal directions or different frequencies. Figure 3 illustrates how the learned features are associated with the other ones, using iconified representations. We can see: 1) associations with positive weights between features were quite spatially-localized and occur particularly with similar orientations, and 2) those with negative weights especially occur from cross-oriented features to a target, which were sometimes non-localized and overlapped to the target feature. Notice that in the DN transform (7), these positive weights learned in the SEM perform as inhibitory and will suppress the energies of the filters having similar properties.

### 4.3 Magnetoencephalography (MEG)

Brain activity was recorded in a single healthy subject who received alternating visual, auditory, and tactile stimulation interspersed with rest periods [25]. The original signals were measured in $204$ channels (sensors) for several minutes with sampling rate (75Hz); the total number of measurements, i.e. sample size, was $N = 73,760$. As a preprocessing, we applied a band-pass filter (8-30Hz) and remove some outliers. Also, we subtracted the sample mean and then reduced the dimensionality by PCA to $d = 24$, with $90\%$ of variance still retained.

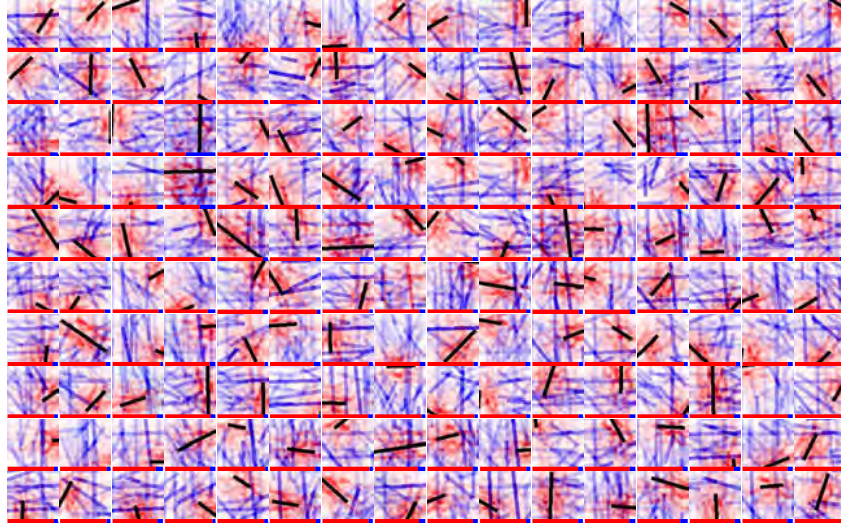

Figure 3: Depiction of connection properties between learned basis functions in a similar manner to that has used in e.g. [6]. In each small panel, the black bar depicts the position, orientation and length of a single Gabor-like basis function obtained by our method; the red (resp. blue) pattern of superimposed bars is a linear combination of the bars of the other basis functions according to the absolute values of positive (resp. negative) connection weights to the target one. The intensities of red and blue colors were adjusted separately from each other in each panel; the ratio of the maximum positive and negative connection strengths is depicted at the bottom of each small panel by the relative length of horizontal color bars.

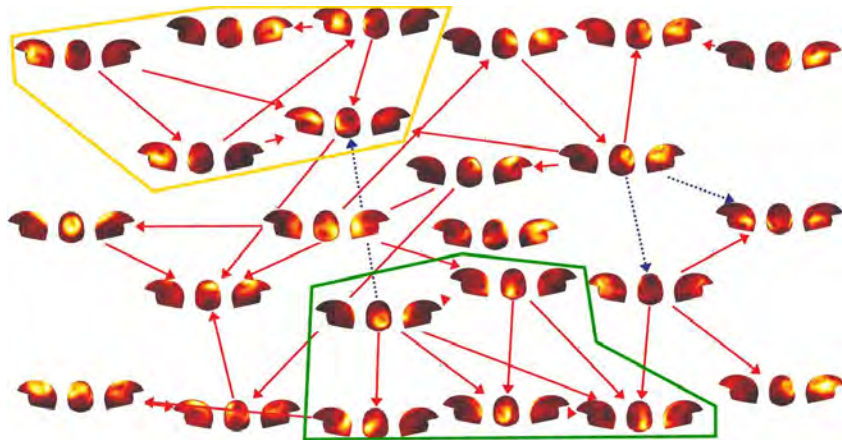

Figure 4: Estimated interaction graph (DAG) for MEG data. The red and blue edges respectively denotes the positive and negative connections. Only the edges with strong connections are drawn, where the absolute threshold value was the same for positive and negative weights. The two manually-inserted contours denote possible clusters of sources (see text).

Figure 4 shows an interaction graph under the DAG constraint. One cluster of components, high-lighted in the figure by the manually inserted yellow contour, seems to consist of components related to auditory processing. The components are located in the temporal cortex, and all but one in the left hemisphere. The direction of influence, which we can estimate in the acyclic model, seems to be from the anterior areas to posterior ones. This may be related to top-down influence, since the primary auditory cortex seems to be included in the posterior areas on the left hemisphere; at the end of the chain, the signal goes to the right hemisphere. Such temporal components are typically quite difficult to find because the modulation of their energies is quite weak. Our method may help in grouping such components together by analyzing the energy correlations.

Another cluster of components consists of low-level visual areas, highlighted by the green contour. It is more difficult to interpret these interactions because the areas corresponding to the components are very close to each other. It seems, however, that here the influences are mainly from the primary visual areas to the higher-order visual areas.

## 5  Conclusion

We proposed a new statistical model that uses SEM to model energy-dependencies of latent variables in a standard linear generative model. In particular, with a simplified form of scale-mixture model, the likelihood function was derived without any approximation. The SEM has both acyclic and cyclic variants. In the acyclic case, non-Gaussianity is essential for identifiability, while in the cyclic case we introduces the constraint of symmetricity which also guarantees identifiability. We also provided a new generative interpretation of DN transform based on a nonlinear SEM. Our method exhibited a high applicability in three simulations each with synthetic dataset, natural images, and brain signals.

## Appendix: Derivation of Eq. (8)

From the uniformity of signs, we have $p_s(s) = p_s(\mathbf{D}s)$ for any $\mathbf{D} = \mathrm{diag}(\pm 1, \dots, \pm 1)$; particularly, let $\mathbf{D}_k$ correspond to the signs of $k$-th orthant $\mathcal{S}_k$ of $\mathbb{R}^d$, and $\mathcal{S}_1 = (0, \infty)^d$. Then, the relation $\int_{\mathcal{S}_1} d\boldsymbol{\sigma}\, p_{\boldsymbol{\sigma}}(\boldsymbol{\sigma}) = \sum_{k=1}^{K} \int_{\mathcal{S}_k} ds\, p_s(s) = \sum_{k=1}^{K} \int_{\mathcal{S}_1} d\boldsymbol{\sigma}\, p_s(\mathbf{D}_k\boldsymbol{\sigma}) = 2^d \int_{\mathcal{S}_1} d\boldsymbol{\sigma}\, p_s(\boldsymbol{\sigma})$ implies $p_s(s) = (1/2^d)p_{\boldsymbol{\sigma}}(s)$ for any $s \in \mathcal{S}_1$; thus $p_s(s) = (1/2^d)p_{\boldsymbol{\sigma}}(|s|)$ for any $s \in \mathbb{R}^d$. Now, $y = \ln \phi(\sigma)$ (for every component) and thus $p_{\boldsymbol{\sigma}}(\boldsymbol{\sigma}) = p_y(y) \prod_i |(\ln \phi)'(\sigma_i)|$, where we assume $\phi$ is differentiable. Let $\psi(s) := \ln \phi(|s|)$ and $\psi'(s) := \mathrm{sign}(s)(\ln \phi)'(|s|)$. Then it follows that $p_s(s) = (1/2^d)p_y(\psi(s)) \prod_i |\psi'(s_i)|$, where $\psi(s)$ performs component-wise. Since $y$ maps linearly to $r$ with the absolute Jacobian $|\det \mathbf{V}|$, we have $p_y(y) = |\det \mathbf{V}| \prod_i \rho(r_i)$; combining it with $p_s$ above, we obtain Eq. (8).

## Acknowledgements

We would like to thank Jesús Malo and Valero Laparra for inspiring this work, Michael Gutmann and Patrik Hoyer for helpful discussions and providing codes for fitting Gabor functions and visualization. The MEG data was kindly provided by Pavan Ramkumar and Riitta Hari. J.H. was partially supported by JSPS Research Fellowships for Young Scientists.

## Footnotes

[1]To be precise, [20] showed the invertibility of the entire mapping $s \mapsto z$ in the case of a "signed" DN transform that keeps the signs of $z_i$ and $s_i$ to be the same.

[2]Under the dynamical system interpretation, the matrix $\mathbf{H}$ should have absolute eigenvalues smaller than one for stability [18], where $\mathbf{V} = \mathbf{I} - \mathbf{H}$ is naturally positive definite because the eigenvalues are all positive.

[3] Matlab package is available at http://research.ics.tkk.fi/ica/fastica/. We used the following options: g=tanh, approach=symm, epsilon=$10^{-6}$, MaxNumIterations=$10^4$, finetune=tanh.

[4] Available in Imageica Toolbox by Patrik Hoyer, at http://www.cs.helsinki.fi/u/phoyer/software.html

## References

[1] S. Amari, A. Cichoki, and H. H. Yang. A new learning algorithm for blind signal separation. In *Advances in Neural Information Processing Systems*, volume 8, 1996.

[2] A. J. Bell and T. J. Sejnowski. The 'independent components' of natural scenes are edge filters. *Vision Res.*, 37:3327–3338, 1997.

[3] K. A. Bollen. *Structural Equations with Latent Variables*. Wiley, New York, 1989.

[4] M. Carandini, D. J. Heeger, and J. A. Movshon. Linearity and normalization in simple cells of the macaque primary visual cortex. *Journal of Neuroscience*, 17:8621–8644, 1997.

[5] A. Cichocki and P. Georgiev. Blind source separation algorithms with matrix constraints. *IEICE Trans. Fundamentals*, E86-A(3):522–531, 2003.

[6] P. Garrigues and B. A. Olshausen. Learning horizontal connections in a sparse coding model of natural images. In *Advances in Neural Information Processing Systems*, volume 20, pages 505–512, 2008.

[7] D. J. Heeger. Normalization of cell responses in cat striate cortex. *Visual Neuroscience*, 9:181–197, 1992.

[8] P. O. Hoyer, D. Janzing, J. Mooij, J. Peters, and B. Schölkopf. Nonlinear causal discovery with additive noise models. In *Advances in Neural Information Processing Systems*, volume 21, pages 689–696, 2009.

[9] A. Hyvärinen. Fast and robust fixed-point algorithms for independent component analysis. *IEEE Transactions on Neural Networks*, 10(3):626–634, 1999.

[10] A. Hyvärinen and P.O. Hoyer. Emergence of phase and shift invariant features by decomposition of natural images into independent feature subspaces. *Neural Comput.*, 12(7):1705–1720, 2000.

[11] A. Hyvärinen, P.O. Hoyer, and M. Inki. Topographic independent component analysis. *Neural Comput.*, 13(7):1527–1558, 2001.

[12] A Hyvärinen, J. Hurri, and P. O. Hoyer. *Natural Image Statistics – A probabilistic approach to early computational vision*. Springer-Verlag, 2009.

[13] A. Hyvärinen, J. Karhunen, and E. Oja. *Independent Component Analysis*. John Wiley & Sons, 2001.

[14] Y. Karklin and M. S. Lewicki. A hierarchical Bayesian model for learning nonlinear statistical regularities in nonstationary natural signals. *Neural Comput.*, 17:397–423, 2005.

[15] Y. Karklin and M. S. Lewicki. Emergence of complex cell properties by learning to generalize in natural scenes. *Nature*, 457:83–86, January 2009.

[16] M. Kawanabe and K.-R. Müller. Estimating functions for blind separation when sources have variance dependencies. *Journal of Machine Learning Research*, 6:453–482, 2005.

[17] U. Köster and A. Hyvärinen. A two-layer model of natural stimuli estimated with score matching. *Neural Comput.*, 22:2308–2333, 2010.

[18] G. Lacerda, P. Spirtes, J. Ramsey, and P. Hoyer. Discovering cyclic causal models by independent components analysis. In *Proceedings of the Twenty-Fourth Conference Annual Conference on Uncertainty in Artificial Intelligence (UAI'08)*, pages 366–374, 2008.

[19] S. Lyu. Divisive normalization: Justification and effectiveness as efficient coding transform. In *Advances in Neural Information Processing Systems 23*, pages 1522–1530, 2010.

[20] J. Malo, I. Epifanio, R. Navarro, and E. P. Simoncelli. Nonlinear image representation for efficient perceptual coding. *IEEE Trans Image Process*, 15(1):68–80, 2006.

[21] J. Malo and V. Laparra. Psychophysically tuned divisive normalization approximately factorizes the PDF of natural images. *Neural Comput.*, 22(12):3179–3206, 2010.

[22] B. A. Olshausen and D. J. Field. Emergence of simple-cell receptive field properties by learning a sparse code for natural images. *Nature*, 381:607–609, 1996.

[23] S. Osindero, M. Welling, and G. E. Hinton. Topographic product models applied to natural scene statistics. *Neural Comput.*, 18:381–414, 2006.

[24] J. Pearl. On the statistical interpretation of structural equations. Technical Report R-200, UCLA Cognitive Systems Laboratory, 1993.

[25] P. Ramkumar, L. Parkkonen, R. Hari, and A. Hyvärinen. Characterization of neuromagnetic brain rhythms over time scales of minutes using spatial independent component analysis. *Human Brain Mapping*, 2011. In press.

[26] O. Schwartz and E. P. Simoncelli. Natural signal statistics and sensory gain control. *Nature Neuroscience*, 4(8), 2001.

[27] S. Shimizu, P.O. Hoyer, A. Hyvärinen, and A. Kerminen. A linear non-Gaussian acyclic model for causal discovery. *Journal of Machine Learning Research*, 7:2003–2030, 2006.

[28] E. P. Simoncelli and B. A. Olshausen. Natural image statistics and neural representation. *Annu. Rev. Neurosci.*, 24:1193–1216, 2001.

[29] R. Valerio and R. Navarro. Optimal coding through divisive normalization models of V1 neurons. *Network: Computation in Neural Systems*, 14:579–593, 2003.

[30] H. Valpola, M. Harva, and J. Karhunen. Hierarchical models of variance sources. *Signal Processing*, 84(2):267–282, 2004.

[31] J. H. van Hateren and A. van der Schaaf. Independent component filters of natural images compared with simple cells in primary visual cortex. *Proc. R. Soc. Lond. B*, 265(359–366), 1998.

[32] M. J. Wainwright and E. P. Simoncelli. Scale mixtures of gaussians and the statistics of natural images. In *Advances in Neural Information Processing Systems*, volume 12, pages 855–861, 2000.

[33] K. Zhang and A. Hyvärinen. Source separation and higher-order causal analysis of MEG and EEG. In *Proceedings of the Twenty-Sixth Conference (UAI 2010)*, pages 709–716, 2010.

